# Applications of Neural Networks in Video Signal Processing

**John C. Pearson, Clay D. Spence and Ronald Sverdlove**
David Sarnoff Research Center
CN5300
Princeton, NJ 08543-5300

## Abstract

Although color TV is an established technology, there are a number of longstanding problems for which neural networks may be suited. Impulse noise is such a problem, and a modular neural network approach is presented in this paper. The training and analysis was done on conventional computers, while real-time simulations were performed on a massively parallel computer called the Princeton Engine. The network approach was compared to a conventional alternative, a median filter. Real-time simulations and quantitative analysis demonstrated the technical superiority of the neural system. Ongoing work is investigating the complexity and cost of implementing this system in hardware.

## 1 THE POTENTIAL FOR NEURAL NETWORKS IN CONSUMER ELECTRONICS

Neural networks are most often considered for application in emerging *new* technologies, such as speech recognition, machine vision, and robotics. The fundamental ideas behind these technologies are still being developed, and it will be some time before products containing neural networks are manufactured. As a result, research in these areas will not drive the development of inexpensive neural network hardware which could serve as a catalyst for the field of neural networks in general.

In contrast, neural networks are rarely considered for application in mature technologies, such as consumer electronics. These technologies are based on established principles of information processing and communication, and they are used in millions of products per year. The embedding of neural networks within such mass-

market products would certainly fuel the development of low-cost network hardware, as economics dictates rigorous cost-reduction in every component.

## 2    IMPULSE NOISE IN TV

The color television signaling standard used in the U.S. was adopted in 1953 (McIlwain and Dean, 1956; Pearson, 1975). The video information is first broadcast as an amplitude modulated (AM) radio-frequency (RF) signal, and is then demodulated in the receiver into what is called the composite video signal. The composite signal is comprised of the high-bandwidth (4.2 MHz) luminance (black and white) signal and two low-bandwidth color signals whose amplitudes are modulated in quadrature on a 3.58 MHz subcarrier. This signal is then further decoded into the red, green and blue signals that drive the display. One image "frame" is formed by interlacing two successive "fields" of 262.5 horizontal lines.

Electric sparks create broad-band RF emissions which are transformed into oscillatory waveforms in the composite video signal, called AM impulses. See Figure 1. These impulses appear on a television screen as short, horizontal, multi-colored streaks which clearly stand out from the picture. Such sparks are commonly created by electric motors. There is little spatial (within a frame) or temporal (between frames) correlation between impulses.

General considerations suggest a two step approach for the removal of impulses from the video signal – *detect* which samples have been corrupted, and *replace* them with values derived from their spatio-temporal neighbors. Although impulses are quite visible, they form a small fraction of the data, so only those samples detected as corrupted should be altered. An interpolated average of some sort will generally be a good estimate of impulse-corrupted samples because images are generally smoothly varying in space and time.

There are a number of difficulties associated with this detection/replacement approach to the problem. There are many impulse-like waveforms present in normal video, which can cause "false positives" or "false alarms". See Figure 2. The algorithms that decode the composite signal into RGB spread impulses onto neighboring lines, so it is desirable to remove the impulses in the composite signal. However, the color encoding within the composite signal complicates matters. The subcarrier frequency is near the ringing frequency of the impulses and tends to hide the impulses. Furthermore, the replacement function cannot simply average the nearest

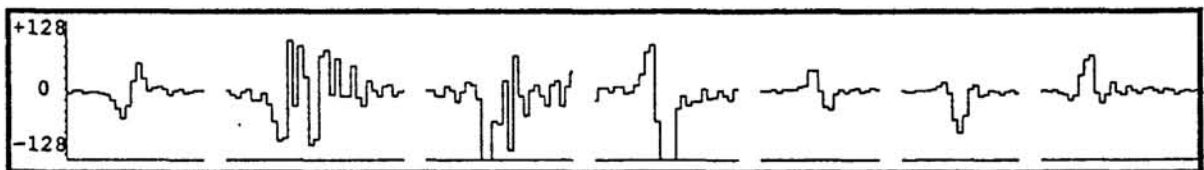

Figure 1: Seven Representative AM Impulse Waveforms. They have been digitized and displayed at the intervals used in digital receivers (8 bits, .07 usec). The largest amplitude impulses are 20-30 samples wide, approximately 3% of the width of one line of active video (752 samples).

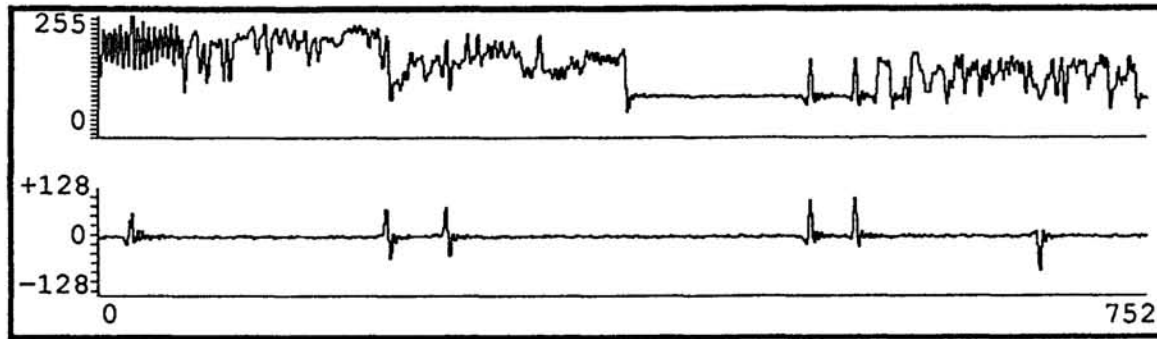

Figure 2: Corrupted Video Scan Line. (Top) Scan line of a composite video signal containing six impulse waveforms. (Bottom) The impulse waveforms, derived by subtracting the uncorrupted signal from the corrupted signal. Note the presence of many impulse-like features in the video signal.

samples, because they represent different color components. The impulses also have a wide variety of waveforms (Figure 1), including some variation caused by clipping in the receiver.

## 3   MODULAR NEURAL NETWORK SYSTEM

The impulse removal system incorporates three small multi-layer perceptron networks (Rumelhart and McClelland, 1986), and all of the processing is confined to one field of data. See Figure 3. The replacement function is performed by one network, termed the i-net ("i" denotes interpolation). Its input is 5 consecutive samples each from the two lines above and the two lines below the current line. The network consists of 10 units in the first hidden layer, 5 in the second, and one output node trained to estimate the center sample of the current line.

The detection function employs 2 networks in series. (A single network detector has been tried, but it has never performed as well as this two-stage detector.) The inputs to the first network are 9 consecutive samples from the current line centered on the sample of interest. It has 3 nodes in the first layer, and one output node trained to compute a moving average of the absolute difference between the clean and noisy signals of the current inputs. It is thus trained to function as a filter for impulse energy, and is termed the e-net. The output of the e-net is then low-pass filtered and sub-sampled to remove redundant information.

The inputs to the second network are 3 lines of 5 consecutive samples each, drawn from the post-processed output of the e-net, centered on the sample of interest. This network, like the e-net, has 3 nodes in the first layer and one output node. It is trained to output 1 if the sample of interest is contaminated with impulse noise, and 0 otherwise. It is thus an impulse detector, and is called the d-net.

The output of the d-net is then fed to a binary switch, which passes through to the final system output either the output of the i-net or the original signal, depending on whether the input exceeds an adjustable threshold.

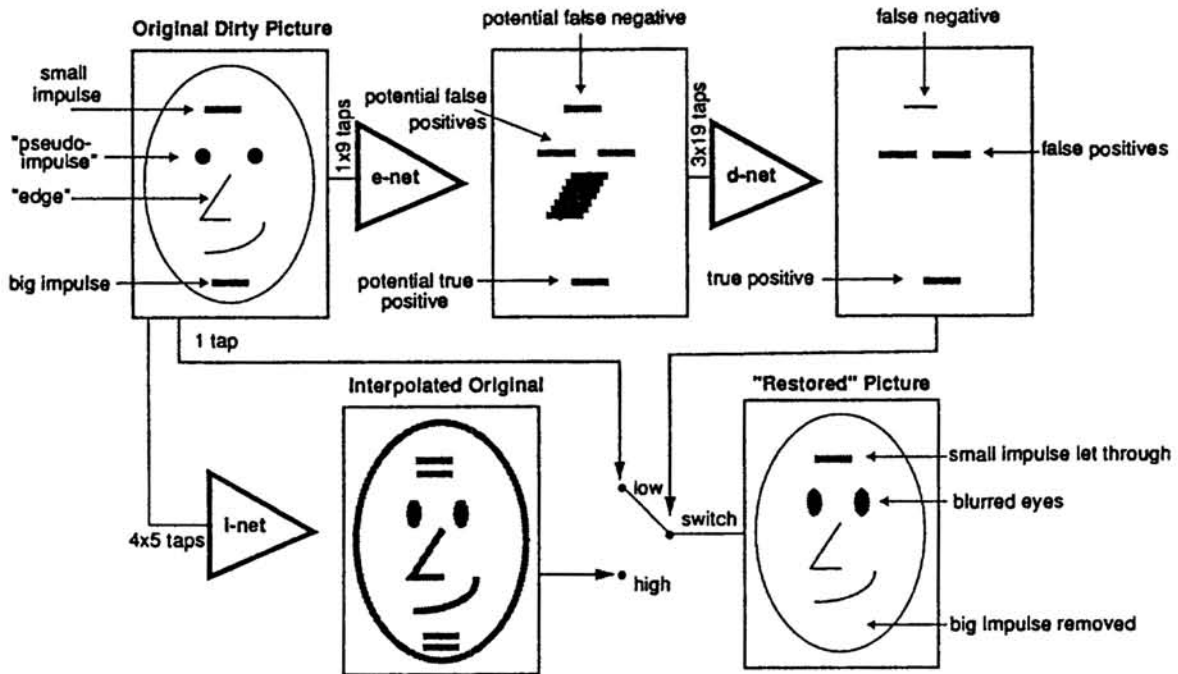

Figure 3: The Neural Network AM Impulse Removal System. The cartoon face is used to illustrate salient image processing characteristics of the system. The e-net correctly signals the presence of the large impulse (chin), misses the small impulse (forehead), and incorrectly identifies edges (nose) and points (eyes) as impulses. The d-net correctly disregards the vertically correlated impulse features (nose) and detects the large impulse (chin), but incorrectly misses the small impulse (forehead) and the non-correlated impulse-like features (eyes). The i-net produces a fuzzy (doubled) version of the original, which is used to replace segments identified as corrupted by the d-net.

Experience showed that the d-net tended to produce narrow spikes in response to impulse-like features of the image. To remove this source of false positives, the output of the d-net is averaged over a 19 sample region centered on the sample of interest. This reduces the peak amplitude of signals due to impulse-like features much more than the broad signals produced by true impulses. An impulse is considered to be present if this smoothed signal exceeds a threshold, the level of which is chosen so as to strike a balance between low false positive rates (high threshold), and high true positive rates (low threshold).

Experience also showed that the fringes of the impulses were not being detected. To compensate for this, sub-threshold d-net output samples are set high if they are within 9 samples of a super-threshold d-net sample. Figure 4 shows the output of the resulting trained system for one scan line.

The detection networks were trained on one frame of video containing impulses of 5 different amplitudes with the largest twenty times the smallest. Visually, these ranged from non-objectionable to brightly colored. Standard incremental back-propagation and conjugate gradient (NAG, 1990) were the training proceedures used. The complexity of the e-net and d-net were reduced in phases. These nets

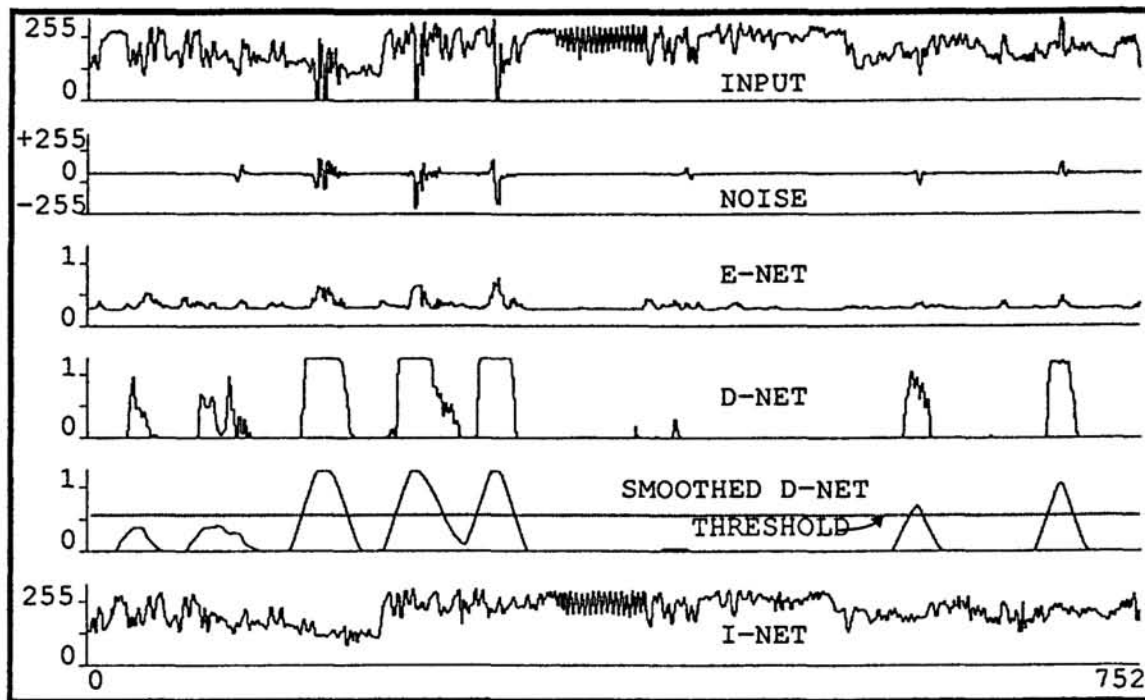

Figure 4: Input and Network Signals.

began as 3 layer nets. After a phase of training, redundant nodes were identified and removed, and training re-started. This process was repeated until there were no redundant nodes.

## 4   REAL-TIME SIMULATION ON THE PRINCETON ENGINE

The trained system was simulated in real-time on the Princeton Engine (Chin et. al., 1988), and a video demonstration was presented at the conference. The Princeton Engine (PE) is a 29.3 GIPS image processing system consisting of up to 2048 processing elements in a SIMD configuration. Each processor is responsible for the output of one column of pixels, and contains a 16-bit arithmetic unit, multiplier, a 64-word triple-port register stack, and 16,000 words of local processor memory. In addition, an interprocessor communication bus permits exchanges of data between neighboring processors during one instruction cycle.

While the i-net performs better than conventional interpolation methods, the difference is not significant for this problem because of the small amount of signal which is replaced. (If the whole image is replaced, the neural net interpolator gave about 1.5 dB better performance than a conventional method.) Thus it has not been implemented on the PE. The i-net may be of value in other video tasks, such as converting from an interlaced to a non-interlaced display.

16-bit fixed point arithmetic was used in these simulations, with 8 bits of fraction, and 10 bit sigmoid function look-up tables. Comparison with the double-precision arithmetic used on the conventional computers showed no significant reduction in

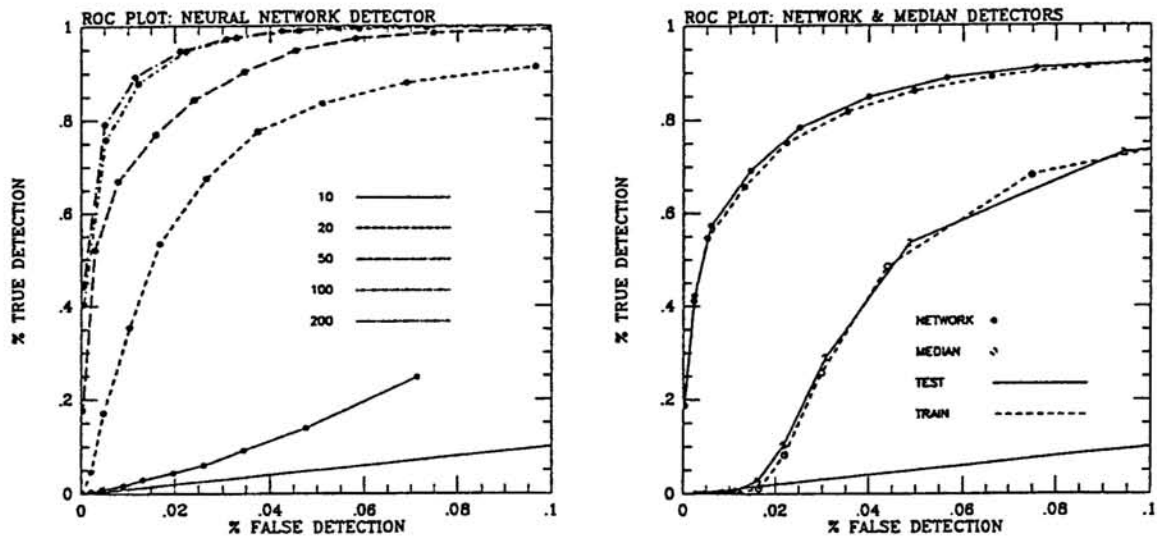

Figure 5: ROC Analysis of Neural Network and Median Detectors.

performance. Current work is exploring the feasibility of implementing training on the PE.

## 5   PERFORMANCE ANALYSIS

The mean squared error (MSE) is well known to be a poor measure of subjective image quality (Roufs and Bouma, 1980). A better measure of detection performance is given by the receiver operating characteristic, or ROC (Green and Swets, 1966, 1974). The ROC is a parametric plot of the fraction of corrupted samples correctly detected versus the fraction of clean samples that were falsely detected. In this case, the decision threshold for the smoothed output of the d-net was the parameter varied. Figure 5 (left) shows the neural network detector ROC for five different impulse amplitudes (tested on a video frame that it was not trained on). This quantifies the sharp breakdown in performance observed in real-time simulations at low impulse amplitude. This breakdown is not observed in analysis of the MSE.

Median filters are often suggested for impulse removal tasks, and have been applied to the removal of impulses from FM TV transmission systems (Perlman, et al, 1987). In order to assess the relative merits of the neural network detector, a median detector was designed and analyzed. This detector computes the median of the current sample and its 4 nearest neighbors with the same color sub-carrier phase. A detection is registered if the difference between the median and the current sample is above threshold (the same additional measures were taken to insure that impulse fringes were detected as were described above for the neural network detector). Figure 5 (right) shows both the neural network and median detector ROC's for two different video frames, each of which contained a mixture of all 5 impulse amplitudes. One frame was used in training the network (TRAIN), and the other was not (TEST). This verifies that the network was not overtrained, and quantifies the superior performance of the network detector observed in real-time simulations.

# 6   CONCLUSIONS

We have presented a system using neural network algorithms that outperforms a conventional method, median filtering, in removing AM impulses from television signals. Of course an additional essential criterion is the cost and complexity of hardware implementations. Median filter chips have been successfully fabricated (Christopher et al., 1988). We are currently investigating the feasibility of casting small neural networks into special purpose chips. We are also applying neural nets to other television signal processing problems.

## Acknowledgements

This work was supported by Thomson Consumer Electronics, under Erich Geiger and Dietrich Westerkamp. This work was part of a larger team effort, and we acknowledge their help, in particular: Nurit Binenbaum, Jim Gibson, Patrick Hsieh, and John Ju.

## References

Chin, D., J. Passe, F. Bernard, H. Taylor and S. Knight, (1988). The Princeton Engine: A Real-Time Video System Simulator. *IEEE Transactions on Consumer Electronics* **34**:2 pp. 285–297.

Christopher, L.A., W.T. Mayweather III, and S. Perlman, (1988). A VLSI Median Filter for Impulse Noise Elimination in Composite or Component TV Signals. *IEEE Transactions on Consumer Electronics* **34**:1 p. 262.

Green, D.M., and J.A. Swets, (1966 and 1974). *Signal Detection Theory and Psychophysics*. New York, Wiley (1966). Reprinted with corrections, Huntington, N.Y., Krieger (1974).

McIlwain, K. and C.E. Dean (eds.); Hazeltine Corporation Staff, (1956). *Principles of Color Television*. New York. John Wiley and Sons.

NAG, (1990). *The NAG Fortran Library Manual, Mark 14*. Downers Grove, IL (The Numerical Algorithms Group Inc.).

Pearson, D.E., (1975). *Transmission and Display of Pictorial Information*. New York. John Wiley and Sons.

Perlman, S.S, S. Eisenhandler, P.W. Lyons, and M.J. Shumila, (1987). Adaptive Median Filtering for Impulse Noise Elimination in Real-Time TV Signals. *IEEE Transactions on Communications* **COM-35**:6 p. 646.

Roufs, J.A. and H. Bouma, (1980). Towards Linking Perception Research and Image Quality. *Proceedings of the SID* **21**:3, pp. 247–270.

Rumelhart, D.E. and J.L. McClelland (eds.), (1986). *Parallel Distributed Processing: Explorations in the Microstructure of Cognition*. Cambridge, Mass., MIT Press.


